# Fast Online Policy Gradient Learning with SMD Gain Vector Adaptation

**Nicol N. Schraudolph      Jin Yu      Douglas Aberdeen**
Statistical Machine Learning, National ICT Australia, Canberra
{nic.schraudolph,douglas.aberdeen}@nicta.com.au

## Abstract

Reinforcement learning by direct policy gradient estimation is attractive in theory but in practice leads to notoriously ill-behaved optimization problems. We improve its robustness and speed of convergence with stochastic meta-descent, a gain vector adaptation method that employs fast Hessian-vector products. In our experiments the resulting algorithms outperform previously employed online stochastic, offline conjugate, and natural policy gradient methods.

## 1   Introduction

Policy gradient reinforcement learning (RL) methods train controllers by estimating the gradient of a long-term reward measure with respect to the parameters of the controller [1]. The advantage of policy gradient methods, compared to value-based RL, is that we avoid the often redundant step of accurately estimating a large number of values. Policy gradient methods are particularly appealing when large state spaces make representing the exact value function infeasible, or when partial observability is introduced. However, in practice policy gradient methods have shown slow convergence [2], not least due to the stochastic nature of the gradients being estimated.

The *stochastic meta-descent* (SMD) gain adaptation algorithm [3, 4] can considerably accelerate the convergence of stochastic gradient descent. In contrast to other gain adaptation methods, SMD copes well not only with stochasticity, but also with non-i.i.d. sampling of observations, which necessarily occurs in RL. In this paper we derive SMD in the context of policy gradient RL, and obtain over an order of magnitude improvement in convergence rate compared to previously employed policy gradient algorithms.

## 2   Stochastic Meta-Descent

### 2.1   Gradient-based gain vector adaptation

Let $R$ be a scalar objective function we wish to maximize with respect to its adaptive parameter vector $\boldsymbol{\theta} \in \mathbb{R}^n$, given a sequence of observations $\boldsymbol{x}_t \in \mathcal{X}$ at time $t = 1, 2, \ldots$ Where $R$ is not available or expensive to compute, we use the *stochastic approximation* $R_t : \mathbb{R}^n \times \mathcal{X} \to \mathbb{R}$ of $R$ instead, and maximize the expectation $\mathbb{E}_t[R_t(\boldsymbol{\theta}_t, \boldsymbol{x}_t)]$. Assuming that $R_t$ is twice differentiable wrt. $\boldsymbol{\theta}$, with gradient and Hessian given by

$$\boldsymbol{g}_t = \tfrac{\partial}{\partial \boldsymbol{\theta}} R_t(\boldsymbol{\theta}, \boldsymbol{x}_t)|_{\boldsymbol{\theta}=\boldsymbol{\theta}_t} \quad \text{and} \quad \boldsymbol{H}_t = \tfrac{\partial^2}{\partial \boldsymbol{\theta}\, \partial \boldsymbol{\theta}^\top} R_t(\boldsymbol{\theta}, \boldsymbol{x}_t)|_{\boldsymbol{\theta}=\boldsymbol{\theta}_t}, \tag{1}$$

respectively, we maximize $\mathbb{E}_t[R_t(\boldsymbol{\theta})]$ by the stochastic gradient ascent

$$\boldsymbol{\theta}_{t+1} = \boldsymbol{\theta}_t + \boldsymbol{\gamma}_t \cdot \boldsymbol{g}_t \,, \tag{2}$$

where $\cdot$ denotes element-wise (Hadamard) multiplication. The gain vector $\boldsymbol{\gamma}_t \in (\mathbb{R}^+)^n$ serves as a diagonal conditioner, providing each element of $\boldsymbol{\theta}$ with its own positive gradient step size. We adapt $\boldsymbol{\gamma}$ by a simultaneous meta-level gradient ascent in the objective $R_t$. A straightforward implementation of this idea is the *delta-delta* algorithm [5], which would update $\boldsymbol{\gamma}$ via

$$\boldsymbol{\gamma}_{t+1} = \boldsymbol{\gamma}_t + \mu \frac{\partial R_{t+1}(\boldsymbol{\theta}_{t+1})}{\partial \boldsymbol{\gamma}_t} = \boldsymbol{\gamma}_t + \mu \frac{\partial R_{t+1}(\boldsymbol{\theta}_{t+1})}{\partial \boldsymbol{\theta}_{t+1}} \cdot \frac{\partial \boldsymbol{\theta}_{t+1}}{\partial \boldsymbol{\gamma}_t} = \boldsymbol{\gamma}_t + \mu \boldsymbol{g}_{t+1} \cdot \boldsymbol{g}_t \,, \tag{3}$$

where $\mu \in \mathbb{R}$ is a scalar meta-step size. In a nutshell, gains are decreased where a negative autocorrelation of the gradient indicates oscillation about a local minimum, and increased otherwise. Unfortunately such a simplistic approach has several problems: Firstly, (3) allows gains to become negative. This can be avoided by updating $\boldsymbol{\gamma}$ multiplicatively, *e.g.* via the *exponentiated gradient* algorithm [6].

Secondly, delta-delta's cure is worse than the disease: individual gains are meant to address ill-conditioning, but (3) actually squares the condition number. The autocorrelation of the gradient must therefore be normalized before it can be used. A popular (if extreme) form of normalization is to consider only the sign of the autocorrelation. Such sign-based methods [5, 7–9], however, do not cope well with stochastic approximation of the gradient since the non-linear sign function does not commute with the expectation operator [10]. More recent algorithms [3, 4, 10] therefore use multiplicative (hence linear) normalization factors to condition the meta-level update.

Finally, (3) fails to take into account that gain changes affect not only the current, but also future parameter updates. In recognition of this shortcoming, $\boldsymbol{g}_t$ in (3) is often replaced with a running average of past gradients. Though such ad-hoc smoothing does improve performance, it does not properly capture long-term dependences, the average still being one of immediate, single-step effects. By contrast, Sutton [11] modeled the long-term effect of gains on future parameter values in a linear system by carrying the relevant partials forward in time, and found that the resulting gain adaptation can outperform a less than perfectly matched Kalman filter. Stochastic meta-descent (SMD) extends this approach to arbitrary twice-differentiable nonlinear systems, takes into account the full Hessian instead of just the diagonal, and applies a decay to the partials being carried forward.

## 2.2 The SMD Algorithm

SMD employs two modifications to address the problems described above: it adjusts gains in log-space, and optimizes over an exponentially decaying trace of gradients. Thus $\ln \boldsymbol{\gamma}$ is updated as follows:

$$\ln \boldsymbol{\gamma}_{t+1} = \ln \boldsymbol{\gamma}_t + \mu \sum_{i=0}^{t} \lambda^i \frac{\partial R(\boldsymbol{\theta}_{t+1})}{\partial \ln \boldsymbol{\gamma}_{t-i}}$$

$$= \ln \boldsymbol{\gamma}_t + \mu \frac{\partial R(\boldsymbol{\theta}_{t+1})}{\partial \boldsymbol{\theta}_{t+1}} \cdot \sum_{i=0}^{t} \lambda^i \frac{\partial \boldsymbol{\theta}_{t+1}}{\partial \ln \boldsymbol{\gamma}_{t-i}} =: \ln \boldsymbol{\gamma}_t + \mu \, \boldsymbol{g}_{t+1} \cdot \boldsymbol{v}_{t+1}, \tag{4}$$

where the vector $\boldsymbol{v} \in \mathbb{R}^n$ characterizes the long-term dependence of the system parameters on their gain history over a time scale governed by the decay factor $0 \leq \lambda \leq 1$. Element-wise exponentiation of (4) yields the desired multiplicative update

$$\boldsymbol{\gamma}_{t+1} = \boldsymbol{\gamma}_t \cdot \exp(\mu \, \boldsymbol{g}_{t+1} \cdot \boldsymbol{v}_{t+1}) \approx \boldsymbol{\gamma}_t \cdot \max(\tfrac{1}{2}, 1 + \mu \, \boldsymbol{g}_{t+1} \cdot \boldsymbol{v}_{t+1}). \tag{5}$$

The linearization $e^u \approx \max(\tfrac{1}{2}, 1 + u)$ eliminates an expensive exponentiation for each gain update, improves its robustness by reducing the effect of outliers ($|u| \gg 0$), and ensures

that $\boldsymbol{\gamma}$ remains positive. To compute the gradient trace $\boldsymbol{v}$ efficiently, we expand $\boldsymbol{\theta}_{t+1}$ in terms of its recursive definition (2):

$$\boldsymbol{v}_{t+1} = \sum_{i=0}^{t} \lambda^i \frac{\partial \boldsymbol{\theta}_{t+1}}{\partial \ln \boldsymbol{\gamma}_{t-i}} = \sum_{i=0}^{t} \lambda^i \frac{\partial \boldsymbol{\theta}_t}{\partial \ln \boldsymbol{\gamma}_{t-i}} + \sum_{i=0}^{t} \lambda^i \frac{\partial (\boldsymbol{\gamma}_t \cdot \boldsymbol{g}_t)}{\partial \ln \boldsymbol{\gamma}_{t-i}} \qquad (6)$$

$$\approx \lambda \boldsymbol{v}_t + \boldsymbol{\gamma}_t \cdot \boldsymbol{g}_t + \boldsymbol{\gamma}_t \cdot \left[ \frac{\partial \boldsymbol{g}_t}{\partial \boldsymbol{\theta}_t} \sum_{i=0}^{t} \lambda^i \frac{\partial \boldsymbol{\theta}_t}{\partial \ln \boldsymbol{\gamma}_{t-i}} \right]$$

Noting that $\frac{\partial \boldsymbol{g}_t}{\partial \boldsymbol{\theta}_t}$ is the Hessian $\boldsymbol{H}_t$ of $R_t(\boldsymbol{\theta}_t)$, we arrive at the simple iterative update

$$\boldsymbol{v}_{t+1} = \lambda \boldsymbol{v}_t + \boldsymbol{\gamma}_t \cdot (\boldsymbol{g}_t + \lambda \boldsymbol{H}_t \boldsymbol{v}_t); \quad \boldsymbol{v}_0 = \boldsymbol{0}. \qquad (7)$$

Although the Hessian of a system with $n$ parameters has $O(n^2)$ entries, efficient indirect methods from algorithmic differentiation are available to compute its product with an arbitrary vector in the same time as 2–3 gradient evaluations [12, 13]. To improve stability, SMD employs an extended Gauss-Newton approximation of $\boldsymbol{H}_t$ for which a similar (even faster) technique is available [4]. An iteration of SMD — comprising (5), (2), and (7) — thus requires less than 3 times the floating-point operations of simple gradient ascent. The extra computation is typically more than compensated for by the faster convergence of SMD. Fast convergence minimizes the number of expensive world interactions required, which in RL is typically of greater concern than computational cost.

## 3 Policy Gradient Reinforcement Learning

A Markov decision process (MDP) consists of a finite[1] set of states $s \in \mathcal{S}$ of the world, actions $a \in \mathcal{A}$ available to the agent in each state, and a (possibly stochastic) reward function $r(s)$ for each state $s$. In a partially observable MDP (POMDP), the controller sees only an observation $\boldsymbol{x} \in \mathcal{X}$ of the current state, sampled stochastically from an unknown distribution $\mathbb{P}(\boldsymbol{x}|s)$. Each action $a$ determines a stochastic matrix $\boldsymbol{P}(a) = [\mathbb{P}(s'|s, a)]$ of transition probabilities from state $s$ to state $s'$ given action $a$. The methods discussed in this paper do not assume explicit knowledge of $\boldsymbol{P}(a)$ or of the observation process. All policies are stochastic, with a probability of choosing action $a$ given state $s$, and parameters $\boldsymbol{\theta} \in \mathbb{R}^n$ of $\mathbb{P}(a|\boldsymbol{\theta}, s)$. The evolution of the state $s$ is Markovian, governed by an $|\mathcal{S}| \times |\mathcal{S}|$ transition probability matrix $\boldsymbol{P}(\boldsymbol{\theta}) = [\mathbb{P}(s'|\boldsymbol{\theta}, s)]$ with entries given by

$$\mathbb{P}(s'|\boldsymbol{\theta}, s) = \sum_{a \in \mathcal{A}} \mathbb{P}(a|\boldsymbol{\theta}, s) \, \mathbb{P}(s'|s, a). \qquad (8)$$

### 3.1 GPOMDP Monte Carlo estimates of gradient and hessian

GPOMDP is an infinite-horizon policy gradient method [1] to compute the gradient of the *long-term average reward*

$$R(\boldsymbol{\theta}) := \lim_{T \to \infty} \frac{1}{T} \mathbb{E}_{\boldsymbol{\theta}} \left[ \sum_{t=1}^{T} r(s_t) \right], \qquad (9)$$

with respect the policy parameters $\boldsymbol{\theta}$. The expectation $\mathbb{E}_{\boldsymbol{\theta}}$ is over the distribution of state trajectories $\{s_0, s_1, \dots\}$ induced by $\boldsymbol{P}(\boldsymbol{\theta})$.

**Theorem 1 (1)** *Let $\boldsymbol{I}$ be the identity matrix, and $\boldsymbol{u}$ a column vector of ones. The gradient of the long-term average reward wrt. a policy parameter $\theta_i$ is*

$$\nabla_{\theta_i} R(\boldsymbol{\theta}) = \boldsymbol{\pi}(\boldsymbol{\theta})^\top \nabla_{\theta_i} \boldsymbol{P}(\boldsymbol{\theta}) [\boldsymbol{I} - \boldsymbol{P}(\boldsymbol{\theta}) + \boldsymbol{u}\boldsymbol{\pi}(\boldsymbol{\theta})^\top]^{-1} \boldsymbol{r}, \qquad (10)$$

*where $\boldsymbol{\pi}(\boldsymbol{\theta})$ is the stationary distribution of states induced by $\boldsymbol{\theta}$.*

Note that (10) requires knowledge of the underlying transition probabilities $\boldsymbol{P}(\boldsymbol{\theta})$, and the inversion of a potentially large matrix. The GPOMDP algorithm instead computes a Monte-Carlo approximation of (10): the agent interacts with the environment, producing an observation, action, reward sequence $\{\boldsymbol{x}_1, a_1, r_1, \boldsymbol{x}_2, \ldots, \boldsymbol{x}_T, a_T, r_T\}$.[2] Under mild technical assumptions, including ergodicity and bounding all the terms involved, Baxter and Bartlett [1] obtain

$$\widehat{\nabla}_{\boldsymbol{\theta}} R = \frac{1}{T} \sum_{t=0}^{T-1} \nabla_{\boldsymbol{\theta}} \ln \mathbb{P}(a_t|\boldsymbol{\theta}, s_t) \sum_{\tau=t+1}^{T} \beta^{\tau-t-1} r(s_\tau), \tag{11}$$

where a discount factor $\beta \in [0, 1)$ implicitly assumes that rewards are exponentially more likely to be due to recent actions. Without it, rewards would be assigned over a potentially infinite horizon, resulting in gradient estimates with infinite variance. As $\beta$ decreases, so does the variance, but the bias of the gradient estimate increases [1]. In practice, (11) is implemented efficiently via the discounted *eligibility trace*

$$\boldsymbol{e}_t = \beta \boldsymbol{e}_{t-1} + \boldsymbol{\delta}_t, \quad \text{where} \quad \boldsymbol{\delta}_t := \nabla_{\boldsymbol{\theta}} \mathbb{P}(a_t|\boldsymbol{\theta}, s_t) / \mathbb{P}(a_t|\boldsymbol{\theta}, s_t). \tag{12}$$

Now $\boldsymbol{g}_t = r_t \boldsymbol{e}_t$ is the gradient of $R(\boldsymbol{\theta})$ arising from assigning the instantaneous reward to all log action gradients, where $\beta$ gives exponentially more credit to recent actions. Likewise, Baxter and Bartlett [1] give the Monte Carlo estimate of the Hessian as $\boldsymbol{H}_t = r_t(\boldsymbol{E}_t + \boldsymbol{e}_t \boldsymbol{e}_t^\top)$, using an eligibility trace *matrix*

$$\boldsymbol{E}_t = \beta \boldsymbol{E}_{t-1} + \boldsymbol{G}_t - \boldsymbol{\delta}_t \boldsymbol{\delta}_t^\top, \quad \text{where} \quad \boldsymbol{G}_t := \nabla_{\boldsymbol{\theta}}^2 \mathbb{P}(a_t|\boldsymbol{\theta}, s_t) / \mathbb{P}(a_t|\boldsymbol{\theta}, s_t). \tag{13}$$

Maintaining $\boldsymbol{E}$ would be $O(n^2)$, thus computationally expensive for large policy parameter spaces. Noting that SMD only requires the product of $\boldsymbol{H}_t$ with a vector $\boldsymbol{v}$, we instead use

$$\boldsymbol{H}_t \boldsymbol{v} = r_t[\boldsymbol{d}_t + \boldsymbol{e}_t(\boldsymbol{e}_t^\top \boldsymbol{v})], \quad \text{where} \quad \boldsymbol{d}_t = \beta \boldsymbol{d}_{t-1} + \boldsymbol{G}_t \boldsymbol{v} - \boldsymbol{\delta}_t(\boldsymbol{\delta}_t^\top \boldsymbol{v}) \tag{14}$$

is an eligibility trace vector that can be maintained in $O(n)$. We describe the efficient computation of $\boldsymbol{G}_t \boldsymbol{v}$ in (14) for a specific action selection method in Section 3.3 below.

### 3.2 GPOMDP-Based optimization algorithms

Baxter et al. [2] proposed two optimization algorithms using GPOMDP's policy gradient estimates $\boldsymbol{g}_t$: OLPOMDP is a simple online stochastic gradient descent (2) with scalar gain $\gamma_t$. Alternatively, CONJPOMDP performs Polak-Ribière conjugation of search directions, using a noise-tolerant line search to find the approximately best scalar step size in a given search direction. Since conjugate gradient methods are very sensitive to noise [14], CONJPOMDP must average $\boldsymbol{g}_t$ over many steps to obtain a reliable gradient measurement; this makes the algorithm inherently inefficient (*cf.* Section 4).

OLPOMDP, on the other hand, is robust to noise but converges only very slowly. We can, however, employ SMD's gain vector adaptation to greatly accelerate it while retaining the benefits of high noise tolerance and online learning. Experiments (Section 4) show that the resulting SMDPOMDP algorithm can greatly outperform OLPOMDP and CONJPOMDP.

Kakade [15] has applied natural gradient [16] to GPOMDP, premultiplying the policy gradient by the inverse of the online estimate

$$\boldsymbol{F}_t = (1 - \tfrac{1}{t})\boldsymbol{F}_{t-1} + \tfrac{1}{t}(\boldsymbol{\delta}_t \boldsymbol{\delta}_t^\top + \epsilon \boldsymbol{I}) \tag{15}$$

of the Fisher information matrix for the parameter update: $\boldsymbol{\theta}_{t+1} = \boldsymbol{\theta}_t + \boldsymbol{\gamma}_0 \cdot r_t \boldsymbol{F}_t^{-1} \boldsymbol{e}_t$. This approach can yield very fast convergence on small problems, but in our experience does not scale well at all to larger, more realistic tasks; see our experiments in Section 4.

### 3.3 Softmax action selection

For discrete action spaces, a vector of action probabilities $z_t := \mathbb{P}(a_t|y_t)$ can be generated from the output $y_t := f(\theta_t, x_t)$ of a parameterised function $f : \mathbb{R}^n \times \mathcal{X} \to \mathbb{R}^{|\mathcal{A}|}$ (such as a neural network) via the *softmax* function:

$$z_t := \text{softmax}(y_t) = \frac{e^{y_t}}{\sum_{m=1}^{|\mathcal{A}|}[e^{y_t}]_m.} \tag{16}$$

Given action $a_t \sim z_t$, GPOMDP's instantaneous log-action gradient wrt. $y$ is then

$$\tilde{g}_t := \nabla_y[z_t]_{a_t}/[z_t]_{a_t} = u_{a_t} - z_t, \tag{17}$$

where $u_i$ is the unity vector in direction $i$. The action gradient wrt. $\theta$ is obtained by backpropagating $\tilde{g}_t$ through $f$'s *adjoint* system [13], performing an efficient multiplication by the transposed Jacobian of $f$. The resulting gradient $\delta_t := J_f^\top \tilde{g}_t$ is then accumulated in the eligibility trace (12). GPOMDP's instantaneous Hessian for softmax action selection is

$$\tilde{H}_t := \nabla_y^2[z_t]_{a_t}/[z_t]_{a_t} = (u_{a_t} - z_t)(u_{a_t} - z_t)^\top + z_t z_t^\top - \text{diag}(z_t). \tag{18}$$

It is indefinite but reasonably well-behaved: the Gerschgorin circle theorem can be employed to show that its eigenvalues must all lie in the interval $[-\frac{1}{4}, 2]$. Furthermore, its expectation over possible actions is zero:

$$\mathbb{E}_{z_t}(\tilde{H}_t) = [\text{diag}(z_t) - 2z_t z_t^\top + z_t z_t^\top] + z_t z_t^\top - \text{diag}(z_t) = 0. \tag{19}$$

The extended Gauss-Newton matrix-vector product [4] employed by SMD is then given by

$$G_t v_t := J_f^\top \tilde{H}_t J_f v_t, \tag{20}$$

where the multiplication by the Jacobian of $f$ (resp. its transpose) is implemented efficiently by propagating $v_t$ through $f$'s *tangent linear* (resp. adjoint) system [13].

---

**Algorithm 1** SMDPOMDP with softmax action selection

---

1. Given (a) an ergodic POMDP with observations $x_t \in \mathcal{X}$, actions $a_t \in \mathcal{A}$, bounded rewards $r_t \in \mathbb{R}$, and softmax action selection
   (b) a differentiable parametric map $f : \mathbb{R}^n \times \mathcal{X} \to \mathbb{R}^{|\mathcal{A}|}$ (neural network)
   (c) $f$'s adjoint ($u \to J_f^\top u$) and tangent linear ($v \to J_f v$) maps
   (d) free parameters: $\mu \in \mathbb{R}_+$; $\beta, \lambda \in [0,1]$; $\gamma_0 \in \mathbb{R}_+^n$; $\theta_1 \in \mathbb{R}^n$

2. Initialize in $\mathbb{R}^n$: $e_0 = d_0 = v_0 = 0$

3. For $t = 1$ to $\infty$: (a) interact with POMDP:
   i. observe feature vector $x_t$
   ii. compute $z_t := \text{softmax}(f(\theta_t, x_t))$
   iii. perform action $a_t \sim z_t$
   iv. observe reward $r_t$
   (b) maintain eligibility traces:
   i. $\delta_t := J_f^\top(u_{a_t} - z_t)$
   ii. $p_t := J_f v_t$
   iii. $q_t := (u_{a_t} - z_t)(\delta_t^\top v_t) + z_t(z_t^\top p_t) - z_t \cdot p_t$
   iv. $e_t = \beta e_{t-1} + \delta_t$
   v. $d_t = \beta d_{t-1} + J_f^\top q_t - \delta_t(\delta_t^\top v_t)$
   (c) update SMD parameters:
   i. $\gamma_t = \gamma_{t-1} \cdot \max(\frac{1}{2}, 1 + \mu r_t e_t \cdot v_t)$
   ii. $\theta_{t+1} = \theta_t + r_t \gamma_t \cdot e_t$
   iii. $v_{t+1} = \lambda v_t + r_t \gamma_t \cdot [(1 + \lambda e_t^\top v_t)e_t + \lambda d_t]$

---

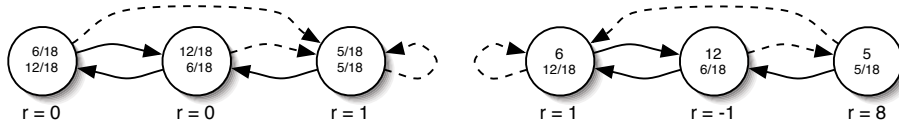

Fig. 1: Left: Baxter et al.'s simple 3-state POMDP. States are labelled with their observable features and instantaneous reward $r$; arrows indicate the 80% likely transition for the first (solid) *resp.* second (dashed) action. Right: our modified, more difficult 3-state POMDP.

## 4 Experiments

### 4.1 Simple Three-State POMDP

Fig. 1 (left) depicts the simple 3-state POMDP used by Baxter et al. [2, Tables 1&2]. Of the two possible transitions from each state, the preferred one occurs with 80% probability, the other with 20%. The preferred transition is determined by the action of a simple probabilistic adaptive controller that receives two state-dependent feature values as input, and is trained to maximize the expected average reward by policy gradient methods.

Using the original code of Baxter et al. [2], we replicated their experimental results for the OLPOMDP and CONJPOMDP algorithms on this simple POMDP. We can accurately reproduce all essential features of their graphed results on this problem [2, Figures 7&8]. We then implemented SMDPOMDP (Algorithm 1), and ran a comparison of algorithms, using the best free parameter settings found by Baxter et al. [2] (in particular: $\beta = 0, \gamma_0 = 1$), and $\mu = \lambda = 1$ for SMDPOMDP. We always match random seeds across algorithms.

Baxter et al. [2] collect and plot results for CONJPOMDP in terms of its $T$ parameter, which specifies the number of Markov chain iterations *per gradient evaluation*. For a fair comparison of convergence speed we added code to record the *total* number of Markov chain iterations consumed by CONJPOMDP, and plot performance for all three algorithms in those terms, with error bars along both axes for CONJPOMDP.

The results are shown in Fig. 2 (left), averaged over 500 runs. While early on CONJPOMDP *on average* reaches a given level of performance about three times faster than OLPOMDP, it does so at the price of far higher variance. Moreover, CONJPOMDP is the only algorithm that fails to asymptotically approach optimal performance ($R = 0.8$; Fig. 2 left, inset). Once its step size adaptation gets going, SMDPOMDP converges asymptotically to the op-

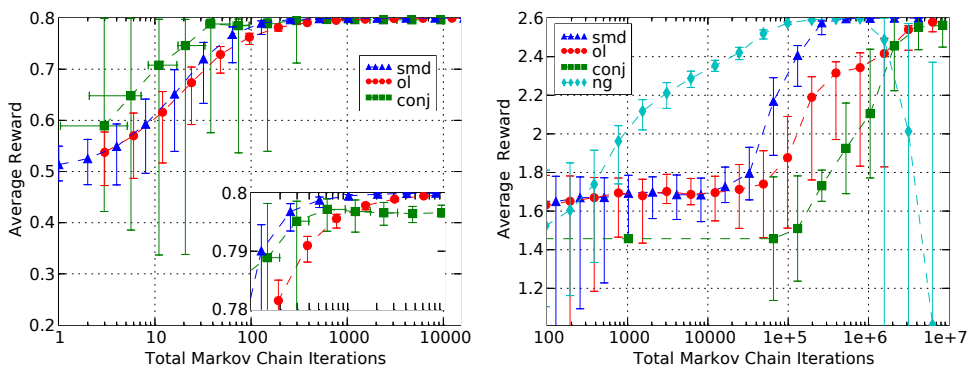

Fig. 2: Left: The POMDP of Fig. 1 (left) is easy to learn. CONJPOMDP converges faster but to asymptotically inferior solutions (see inset) than the two online algorithms. Right: SMDPOMDP outperforms OLPOMDP and CONJPOMDP on the difficult POMDP of Fig. 1 (right). Natural policy gradient has rapid early convergence but diverges asymptotically.

timal policy about three times faster than OLPOMDP in terms of Markov chain iterations, making the two algorithms roughly equal in terms of computational expense.

CONJPOMDP on average performs *less than two iterations* of conjugate gradient in each run. While this is perfectly understandable — the controller only has two trainable parameters — it bears keeping in mind that the performance of CONJPOMDP here is almost entirely governed by the line search rather than the conjugation of search directions.

## 4.2 Modified Three-State POMDP

The three-state POMDP employed by Baxter et al. [2] has the property that greedy maximization of instantaneous reward leads to the optimal policy. Non-trivial temporal credit assignment — the hallmark of reinforcement learning — is not needed. The best results are obtained with the eligibility trace turned off ($\beta = 0$). To create a more challenging problem, we rearranged the POMDP's state transitions and reward structure so that the instantaneous reward becomes deceptive (Fig. 1, right). We also multiplied one state feature by 18 to create an ill-conditioned input to the controller, while leaving the actions and relative transition probabilities (80% resp. 20%) unchanged. In our modified POMDP, the high-reward state can only be reached through an intermediate state with negative reward.

Fig. 2 (right) shows our experimental results for this harder POMDP, averaged over 100 runs. Free parameters were tuned to $\theta_1 \in [-0.1, 0.1]$, $\beta = 0.6$, $\gamma_0 = 0.001$; $T = 10^5$ for CONJPOMDP; $\mu = 0.002$, $\lambda = 1$ for SMDPOMDP. CONJPOMDP now performs the worst, which is expected because conjugation of directions is known to collapse in the presence of noise [14]. SMDPOMDP converges about 20 times faster than OLPOMDP because its adjustable gains compensate for the ill-conditioned input. Kakade's natural gradient (using $\epsilon = 0.01$) performs extremely well early on, taking 2–3 times fewer iterations than SMD-POMDP to reach optimal performance ($R = 2.6$). It does, however, diverge asymptotically.

## 4.3 Puck World

We also implemented the Puck World benchmark of Baxter et al. [2], with the free parameters settings $\theta_1 \in [-0.1, 0.1]$, $\beta = 0.95$ $\gamma_0 = 2 \cdot 10^{-6}$; $T = 10^6$ for CONJPOMDP; $\mu = 100$, $\lambda = 0.999$ for SMDPOMDP; $\epsilon = 0.01$ for natural policy gradient. To improve its stability, we modified SMD here to track instantaneous log-action gradients $\delta_t$ instead of noisy $r_t e_t$ estimates of $\nabla_\theta R$. CONJPOMDP used a quadratic weight penalty of initially 0.5, with the adaptive reduction schedule described by Baxter et al. [2, page 369]; the online algorithms did not require a weight penalty.

Fig. 3 shows our results averaged over 100 runs, except for natural policy gradient where only a single typical run is shown. This is because its $O(n^3)$ time complexity per iteration[3]

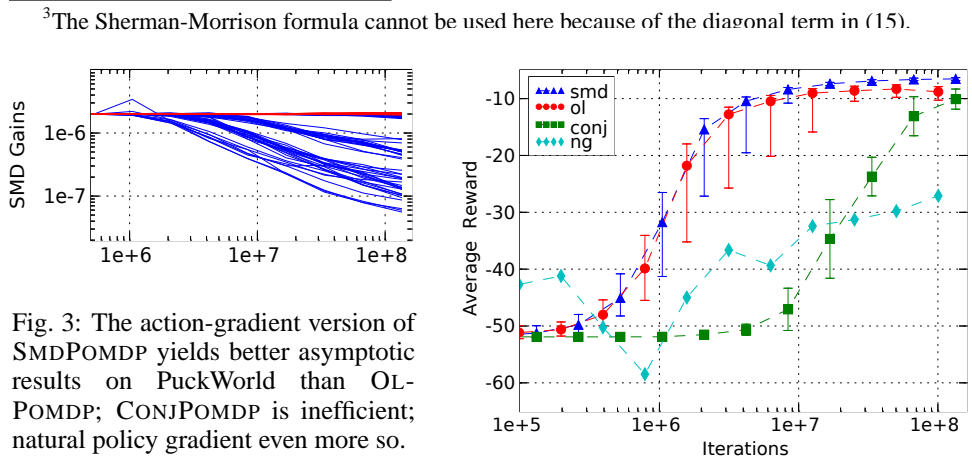

Fig. 3: The action-gradient version of SMDPOMDP yields better asymptotic results on PuckWorld than OL-POMDP; CONJPOMDP is inefficient; natural policy gradient even more so.

makes natural policy gradient intolerably slow for this task, where $n = 88$. Moreover, its convergence is quite poor here in terms of the number of iterations required as well.

CONJPOMDP is again inferior to the best online algorithms by over an order of magnitude. Early on, SMDPOMDP matches OLPOMDP, but then reaches superior solutions with small variance. SMDPOMDP-trained controllers achieve a long-term average reward of -6.5, significantly above the optimum of -8 hypothesized by Baxter et al. [2, page 369] based on their experiments with CONJPOMDP.

## 5  Conclusion

On several non-trivial RL problems we find that our SMDPOMDP consistently outperforms OLPOMDP, which in turn outperforms CONJPOMDP. Natural policy gradient can converge rapidly, but is too unstable and computationally expensive for all but very small controllers.

**Acknowledgements**

We are indebted to John Baxter for his code and helpful comments. National ICT Australia is funded by the Australian Government's Backing Australia's Ability initiative, in part through the Australian Research Council. This work is also supported by the IST Program of the European Community, under the Pascal Network of Excellence, IST-2002-506778.

**References**

[1] J. Baxter and P. L. Bartlett. Infinite-horizon policy-gradient estimation. *Journal of Artificial Intelligence Research*, 15:319–350, 2001.

[2] J. Baxter, P. L. Bartlett, and L. Weaver. Experiments with infinite-horizon, policy-gradient estimation. *Journal of Artificial Intelligence Research*, 15:351–381, 2001.

[3] N. N. Schraudolph. Local gain adaptation in stochastic gradient descent. In *Proc. Intl. Conf. Artificial Neural Networks*, pages 569–574, Edinburgh, Scotland, 1999. IEE, London.

[4] N. N. Schraudolph. Fast curvature matrix-vector products for second-order gradient descent. *Neural Computation*, 14(7):1723–1738, 2002.

[5] R. Jacobs. Increased rates of convergence through learning rate adaptation. *Neural Networks*, 1:295–307, 1988.

[6] J. Kivinen and M. K. Warmuth. Additive versus exponentiated gradient updates for linear prediction. In *Proc. 27th Annual ACM Symposium on Theory of Computing*, pages 209–218. ACM Press, New York, NY, 1995.

[7] T. Tollenaere. SuperSAB: Fast adaptive back propagation with good scaling properties. *Neural Networks*, 3:561–573, 1990.

[8] F. M. Silva and L. B. Almeida. Acceleration techniques for the backpropagation algorithm. In L. B. Almeida and C. J. Wellekens, editors, *Neural Networks: Proc. EURASIP Workshop*, volume 412 of *Lecture Notes in Computer Science*, pages 110–119. Springer Verlag, 1990.

[9] M. Riedmiller and H. Braun. A direct adaptive method for faster backpropagation learning: The RPROP algorithm. In *Proc. Intl. Conf. Neural Networks*, pages 586–591. IEEE, 1993.

[10] L. B. Almeida, T. Langlois, J. D. Amaral, and A. Plakhov. Parameter adaptation in stochastic optimization. In D. Saad, editor, *On-Line Learning in Neural Networks*, Publications of the Newton Institute, chapter 6, pages 111–134. Cambridge University Press, 1999.

[11] R. S. Sutton. Gain adaptation beats least squares? In *Proceedings of the 7th Yale Workshop on Adaptive and Learning Systems*, pages 161–166, 1992.

[12] B. A. Pearlmutter. Fast exact multiplication by the Hessian. *Neural Comput.*, 6(1):147–60, 1994.

[13] A. Griewank. *Evaluating Derivatives: Principles and Techniques of Algorithmic Differentiation*. Frontiers in Applied Mathematics. SIAM, Philadelphia, 2000.

[14] N. N. Schraudolph and T. Graepel. Combining conjugate direction methods with stochastic approximation of gradients. In C. M. Bishop and B. J. Frey, editors, *Proc. 9th Intl. Workshop Artificial Intelligence and Statistics*, pages 7–13, Key West, Florida, 2003.

[15] S. Kakade. A natural policy gradient. In T. G. Dietterich, S. Becker, and Z. Ghahramani, editors, *Advances in Neural Information Processing Systems 14*, pages 1531–1538. MIT Press, 2002.

[16] S. Amari. Natural gradient works efficiently in learning. *Neural Comput.*, 10(2):251–276, 1998.

## Footnotes

[1] For uncountably infinite state spaces, the derivation becomes more complex without substantially altering the resulting algorithms.

[2]We use $r_t$ as shorthand for $r(s_t)$, making it clear that only the reward value is known, not the underlying state $s_t$.

[3] The Sherman-Morrison formula cannot be used here because of the diagonal term in (15).
